# Learning Fine Motion by Markov Mixtures of Experts

**Marina Meilă**
Dept. of Elec. Eng. and Computer Sci.
Massachussetts Inst. of Technology
Cambridge, MA 02139
mmp@ai.mit.edu

**Michael I. Jordan**
Dept.of Brain and Cognitive Sciences
Massachussetts Inst. of Technology
Cambridge, MA 02139
jordan@psyche.mit.edu

## Abstract

Compliant control is a standard method for performing fine manipulation tasks, like grasping and assembly, but it requires estimation of the state of contact (s.o.c.) between the robot arm and the objects involved. Here we present a method to learn a model of the movement from measured data. The method requires little or no prior knowledge and the resulting model explicitly estimates the s.o.c. The current s.o.c. is viewed as the hidden state variable of a discrete HMM. The control dependent transition probabilities between states are modeled as parametrized functions of the measurement. We show that their parameters can be estimated from measurements at the same time as the parameters of the movement in each s.o.c. The learning algorithm is a variant of the EM procedure. The E step is computed exactly; solving the M step exactly is not possible in general. Here, gradient ascent is used to produce an increase in likelihood.

## 1 INTRODUCTION

For a large class of robotics tasks, such as assembly tasks or manipulation of relatively light-weight objects, under appropriate damping of the manipulator the dynamics of the objects can be neglected. For these tasks the main difficulty is in having the robot achieve its goal despite uncertainty in its position relative to the surrounding objects. Uncertainty is due to inaccurate knowledge of the geometric shapes and positions of the objects, of their physical properties (surface friction coefficients), or to positioning errors in the manipulator. The standard solution to this problem is *controlled compliance* first introduced in (Mason, 1981). Under compliant motion, the task is performed in stages; in each stage the robot arm

maintains contact with a selected surface or feature of the environment; the stage ends when contact with the feature corresponding to the next stage is made.

Decomposing the given task into subtasks and specifying each goal or subgoal in terms of contact constraints has proven to be a particularly fertile idea, from which a fair number of approaches have evolved. But each of them have to face and solve the problem of estimating the *state of contact* (i.e. checking if the contact with the correct surface is achieved), a direct consequence of dealing with noisy measurements. Additionally, most approaches assume prior geometrical and physical knowledge of the environment.

In this paper we present a method to *learn* a model of the environment which will serve to estimate the s.o.c. and to predict future positions from noisy measurements. It associates to each state of contact the coresponding *movement model* (m.m.); that is: a relationship between positions, nominal and actual velocities that holds over a domain of the position-nominal velocity space. The current m.m. is viewed as the hidden state variable of a discrete Hidden Markov Model (HMM) with transition probabilities that are parametrized functions of the measurement. We call this model *Markov Mixture of Experts* (MME) and show how its parameters can be estimated. In section 2 the problem is defined, section 3 introduces the learning algorithm, section 4 presents a simulated example and 5 discusses other aspects relevant to the implementation.

## 2  REACHABILITY GRAPHS AND MARKOV MIXTURES OF EXPERTS

For any ensemble of objects, the space of all the relative degrees of freedom of the objects in the ensemble is called the *configuration space* (C-space). Every possible configuration of the ensemble is represented by a unique point in the C-space and movement in the real space maps into continuous trajectories in the C-space (Lozano-Perez, 1983). The sets of points corresponding to each state of contact create a partition over the C-space. Because trajectories are continuous, a point can move from a s.o.c. only to a neighboring s.o.c. This can be depicted by a directed graph with vertices representing states of contact and arcs for the possible transitions between them, called the *reachability graph*. If no constraints on the velocities are imposed, then in the reachability graph each s.o.c. is connected to all its neighbours. But if the range of velocities is restricted, the connectivity of the graph decreases and the connections are generally non-symmetric. Figure 1 shows an example of a C-space and its reachability graph for velocities with only positive components.

Ideally, in the absence of noise, the states of contact can be perfectly observed and every transition through the graph is thus deterministic. To deal with the uncertainty in the measurements, we will attach probabilities to the arcs of the graph in the following way: Let us denote by $Q_i$ the set of configurations corresponding to s.o.c. $i$ and let the movement of a point $x$ with uniform nominal velocity $v$ for a time $\Delta T$ be given by $x(t + \Delta T) = f^*(x, v, \Delta T)$; both $x$ and $v$ are vectors of same dimension as the C-space. Now, let $x'$, $v'$ be the noisy measurements of the true values $x$, $v$, $x \in Q_j$ and $P[x, v|x', v', j]$ the posterior distribution of $(x, v)$ given the measurements and the s.o.c. Then, the probability of transition to a state $i$ from a given state $j$ in time $T_s$ can be expressed as:

$$P[i|x', v', j] = \int_{\{x,v|x \in Q_j, f^*(x,v,T_s) \in Q_i\}} P[x, v|x', v', j] dx\, dv = a_{ij}(x', v') \qquad (1)$$

Defining the transition probability matrix $A = [a_{ji}]_{i,j=1}^m$ and assuming measurement

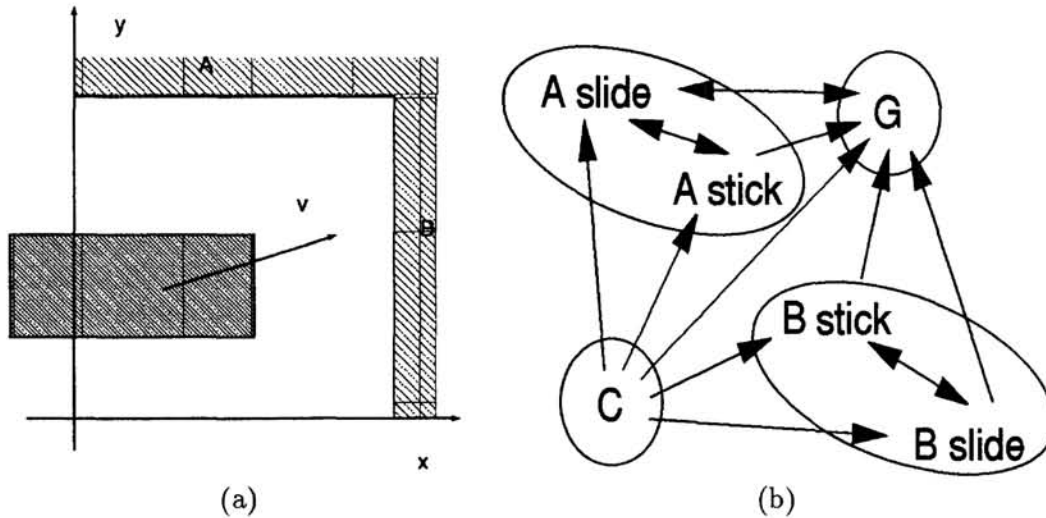

(a)                                        (b)

Figure 1: A configuration space (a) and its reachability graph (b). The nodes represent movement models: C is the free space, A and B are surfaces with static and dynamic friction, G represents jamming in the corner. The velocity V has positive components.

noise $P[x'|q = i, x \in Q_i]$ leads to an HMM with output $x$ having a continuous emission probability distribution and where the s.o.c. plays the role of a hidden state variable. Our main goal is to estimate this model from observed data.

To give a general statement of the problem we will assume that all the position, velocity and force measurements are represented by the input vector $u$; the output vector $y$ of dimensionality $n_y$ contains the future position (which our model will learn to predict). Observations are made at moments which are integer multiples of $T_s$, indexed by $t = 0, 1, .., T$. If $T_s$ is a constant sampling time the dependency of the transition probability on $T_s$ can be ignored. For the purpose of the parameter estimation, the possible dependence between $u(t)$ and $y(t+1)$ will also be ignored, but it should be considered when the trained model is used for prediction.

Throughout the following section we will also assume that the input-output dependence is described by a Gaussian conditional density $p(y(t)|u(t), q(t) = k)$ with mean $f(u(t), \theta_k)$ and variance $\Sigma = \sigma^2 I$. This is equivalent to assuming that given the s.o.c. all noise is additive Gaussian output noise, which is obviously an approximation. But this approximation will allow us to derive certain quantities in closed form in an effective way.

The function $f(u, \theta_k)$ is the m.m. associated with state of contact $k$ (with $\theta_k$ its parameter vector) and $q$ is the selector variable representing it. Sometimes we will find it useful to partition the domain of a m.m. into subdomains and to represent it by a different function (i.e. a different set of parameters $\theta_k$) on each of the subdomains; then, the name *movement model* will be extended to them.

The evolution of $q$ is controlled by a Markov chain which depends on $u$ and of a set of parameters $W$:

$$a_{ij}(u(t), W) = Pr[q(t+1) = i|q(t) = j, u(t)] \quad t = 0, 1, \ldots$$

with

$$\sum_i a_{ij}(u, W) = 1 \quad \forall u, W, j = 1, \ldots, m. \tag{2}$$

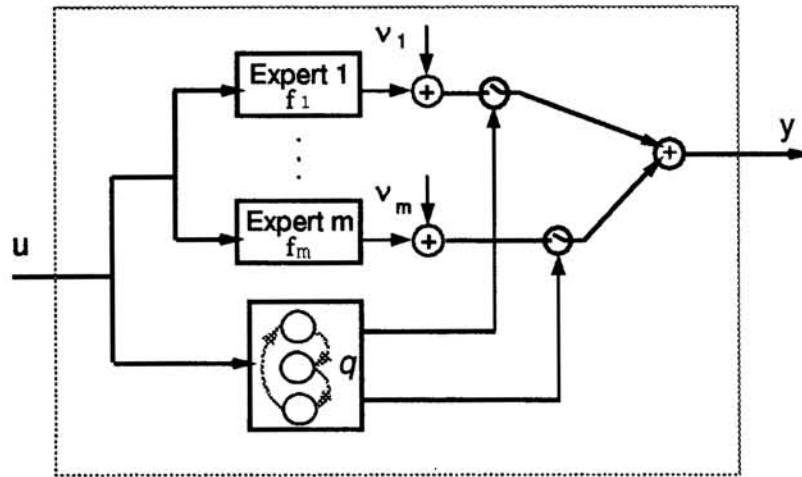

Figure 2: The Markov Mixture of Experts architecture

Fig. 2 depicts this architecture. It can be easily seen that this model generalizes the mixture of experts (ME) architecture (Jacobs, et al., 1991), to which it reduces in the case where $a_{ij}$ are independent of $j$ (the columns of $A$ are all equal). It becomes the model of (Bengio and Frasconi, 1995) when $A$ and $f$ are neural networks.

## 3 AN EM ALGORITHM FOR MME

To estimate the values of the unknown parameters $\sigma^2$, $W_k$, $\theta_k$, $k = 1, \ldots, m$ given the sequence of observations $\{(u(t), y(t))\}_{t=0}^T$, $T > 0$ the *Expectation Maximization* (EM) algorithm will be used. The states $\{q(t)\}_{t=0}^T$ play the role of the unobserved variables. More about EM can be found in (Dempster et al., 1977) while aspects specific to this algorithm are in (Meila and Jordan, 1994).

The E step computes the probability of each state and of every transition to occur at $t \in \{0, \ldots, T\}$ given the observations and an initial parameter set. This can be done efficiently by the *forward-backward* algorithm (Rabiner and Juang, 1986).

$$\gamma_k(t) = Pr[q(t) = k \mid \{(u(t), y(t))\}_{t=0}^T, W, \theta, \sigma^2] \tag{3}$$

$$\xi_{ij}(t) = Pr[q(t) = j, q(t+1) = i \mid \{(u(t), y(t))\}_{t=0}^T, W, \theta, \sigma^2]$$

In the M step the new estimates of the parameters are found by maximizing the average *complete log-likelihood* $J$, which in our case has the form

$$J(\theta, \sigma^2, W) = \sum_{t=0}^{T-1} \sum_{i,j=1}^{m} \xi_{ij}(t) \ln a_{ij}(u(t), W) -$$

$$-\frac{1}{2\sigma^2} \sum_{t=0}^{T} \sum_{k=1}^{m} \gamma_k(t) \|y(t) - f(u(t), \theta_k)\|^2 - \frac{T+1}{2} n_y \ln(\sigma^2) + \text{ct.} \tag{4}$$

Since each parameter appears in only one term of $J$ the maximization is equivalent to:

$$\theta_k^{new} = \underset{\theta_k}{\operatorname{argmin}} \sum_{t=0}^{T} \gamma_k(t) \|y(t) - f(u(t), \theta_k)\|^2 \tag{5}$$

$$W^{new} = \underset{W}{\mathrm{argmax}} \sum_{t=0}^{T-1} \sum_{ij} \xi_{ij}(t) \ln (a_{ij}(u(t), W)) \qquad (6)$$

$$\sigma^{2\,new} = \frac{1}{n_y(T+1)} \sum_{t=0}^{T} \sum_{k=0}^{m} \gamma_k(t) \|y(t) - f(u(t), \theta_k)\|^2 \qquad (7)$$

There is no general closed form solution to (5) and (6). Their difficulty depends on the form of $f$ and $a_{ij}$. The complexity of the m.m. is determined by the geometrical shape of the objects' surfaces. For planar surfaces and no rotational degrees of freedom $f$ is linear in $\theta_k$. Then, (5) becomes a weighted least squares problem which can be solved in closed form.

The functions in $A$ depend both on the movement and of the noise models. Because the noise is propagated through non-linearities to the output, an exact form as in (1) may be hard to compute analytically. Moreover, a correct noise model for each of the possible uncertainties is rarely available (Eberman, 1995). A common practical approach is to trade accuracy for computability and to parametrize $A$ in a form which is easy to update but deprived of physical meaning. In all the cases where maximization cannot be performed exactly, one can resort to *Generalized EM* by merely *increasing* $J$. In particular, gradient ascent in parameter space is a technique which can replace maximization. This modification will not affect the overall convergence of the EM iteration but can significantly reduce its speed.

Because EM only finds *local* maxima of the likelihood, the initialization is important. If $f(u, \theta_k)$ correspond to physical movement models, good initial estimates for their parameters can be available. The same applies to those components of $W$ which bear physical significance. A complementary approach is to reduce the number of parameters by explicitly setting the probabilities of impossible transitions to 0.

## 4   SIMULATION RESULTS

Simulations have been run on the C-space shown in fig. 1. The inputs were the 4-dimensional vectors of position $(x, y)$ and nominal velocity $(V_x, V_y)$; the output was the predicted position. The coordinate range was $[0, 10]$ and the admissible velocities were confined to the upper right quadrant $(V_{max} \geq V_x, V_y \geq V_{min} > 0)$. The restriction in direction implied that the trajectories remain in the coordinate domain; it also appeared in the topology of the reachability graph, which has no transition to the free space from another state.

This model was implemented by a MME. The m.m. are linear in the parameters, corresponding to the piecewise linearity of the true model. To implement the transition matrix $A$ we used a bank of *gating networks*, one for each s.o.c., consisting of 2 layer perceptrons with softmax[1] output. There are 230 free parameters in the gating networks and 64 in the m.m.

The training set included $N = 5000$ data points, in sequences of length $T \leq 6$, all starting in free space. The starting position of the sequence and the nominal velocities at each step were picked randomly. We found that a more uniform distribution of the data points over the states of contact is necessary for successful learning. Since this is not expected to happen in applications (where, e.g., *sticking* occurs less often than *sliding*), the obtained models were tested also on a distribution that

Table 1: Performance of MME versus ME

(a) Model Prediction Standard Error $(MSE)^{1/2}$

| Test set | Training distribution | | | | | Uniform V distribution | | | | |
|---|---|---|---|---|---|---|---|---|---|---|
| noise level | 0 | .1 | .2 | .3 | .4 | 0 | .1 | .2 | .3 | .4 |
| MME, $\sigma$ =.2 | .024 | .113 | .222 | .332 | .443 | .023 | .11 | .219 | .327 | .437 |
| MME, $\sigma$ =0 | .003 | .114 | .228 | .343 | .456 | .010 | .109 | .218 | .327 | .435 |
| ME, $\sigma$ =.2 | .052 | .133 | .25 | .37 | .493 | .044 | .129 | .247 | .367 | .488 |
| ME, $\sigma$ =0 | .047 | .131 | .25 | .37 | .49 | .034 | .126 | .245 | .366 | .488 |

(b) State Misclassification Error [%]

| Test set | Training distribution | | | | | Uniform V distribution | | | | |
|---|---|---|---|---|---|---|---|---|---|---|
| noise level | 0 | .1 | .2 | .3 | .4 | 0 | .1 | .2 | .3 | .4 |
| MME, $\sigma$ =.2 | 5.15 | 5.2 | 5.5 | 5.9 | 6.4 | 3.45 | 3.5 | 3.8 | 4.2 | 4.6 |
| MME, $\sigma$ =0 | .78 | 1.40 | 2.35 | 3.25 | 4.13 | .89 | 1.19 | 1.70 | 2.30 | 2.88 |
| ME, $\sigma$ =.2 | 6.46 | 6.60 | 7.18 | 7.73 | 8.13 | 3.85 | 3.90 | 4.38 | 4.99 | 5.65 |
| ME, $\sigma$ =0 | 6.25 | 6.45 | 6.98 | 7.61 | 8.15 | 3.84 | 3.98 | 4.53 | 5.05 | 5.70 |

was uniform over velocities (and consequently, highly non-uniform over states of contact). Gaussian noise with $\sigma=0.2$ or 0 was added to the $(x, y)$ training data.

In the M step, the parameters of the gating networks were updated by gradient ascent. For the m.m. least squares estimation was used. To ensure that models and gates are correctly coupled, initial values for $\theta$ are chosen around the true values. As discussed in the previous section, this is not an unrealistic assumption. $W$ was initialized with small random values. Each simulation was run until convergence.

We used two criteria to measure the performance of the learning algorithm: square root of prediction MSE and hidden state misclassificaton. The results are summarized in table 1. The test set size is 50,000 in all cases. Input noise is Gaussian with levels between 0 and 0.4. Comparisons were made with a ME model with the same number of states.

The simulations show that the MME architecture is tolerant to input noise, although it is not taking it into account explicitly. The MME consistently outperforms the ME model in both prediction and state estimation accuracy.

## 5   DISCUSSION

An algorithm to estimate the parameters of composite movement models in the presence of noisy measurements has been presented. The algorithm exploits the physical decomposability of the problem and the temporal relationship between the data points to produce estimates of both the model's parameters and the s.o.c. It requires only imprecise initial knowledge about the geometry and physical properties of the system.

**Prediction via MME** The trained model can be used either as an estimator for the state of contact or as a forward model in predicting the next position. For the former goal the forward part of the forward-backward algorithm can be used to implement a recursive estimator or the methods in (Eberman, 1995) can be used. The obtained $\gamma_k(t)$, combined with the outputs of the movement models, will produce a predicted output $\hat{y}$. An improved posterior estimate of $y$ can be obtained

by combining $\hat{y}$ with the current measurement.

**Scaling issues.** Simulations have shown that relatively large datasets are required for training even for a small number of states. But, since the states represent physical entities, the model will inherit the geometrical locality properties thereof. Thus, the number of possible transitions from a state will be bounded by a small constant when the number of states grows, keeping the data complexity linear in $m$.

As a version of EM, our algorithm is batch. It follows that parameters are not adapted on line. In particular, the discretization time $T_s$ must be fixed prior to training. But small changes in $T_s$ can be accounted for by rescaling the velocities $V$. For the other changes, inasmuch as they are local, relearning can be confined to those components of the architecture which are affected.

## Footnotes

[1]The *softmax function* is given by: $\mathrm{softmax}_i(x) = \frac{\exp(W_i^T x)}{\sum_j \exp(W_j^T x)}$, $i = 1, ..m$ with $W_j$, $x$ vectors of the same dimension.

# References

Bengio, Y. and Frasconi, P. (1995). An input output HMM architecture. In G. Tesauro, D. Touretzky, & T. Leen (Eds.), *Neural Information Processing Systems 7*, Cambridge, MA: MIT Press, pp. 427–435.

Dempster, A. P., Laird, N. M., and Rubin, D. B. (1977). Maximum likelihood from incomplete data via the EM algorithm. *Journal of the Royal Statistical Society, B*, 39:1–38.

Eberman, B. S. (1995). *A sequential decision approach to sensing manipulation contact features.* PhD thesis, M.I.T., Dept. of Electrical Engineering.

Jacobs, R. A., Jordan, M. I., Nowlan, S., & Hinton, G. E. (1991). Adaptive mixtures of local experts. *Neural Computation, 3*, 1–12.

Lozano-Perez, T. (1983). Spatial planning: a configuration space approach. *IEEE Transactions on Computers.*

Mason, M. T. (1981). Compliance and force control for computer controlled manipulation. *IEEE Trans. on Systems, Man and Cybernetics.*

Meila, M. and Jordan, M. I. (1994). Learning the parameters of HMMs with auxilliary input. Technical Report 9401, MIT Computational Cognitive Science, Cambridge, MA.

Rabiner, R. L. and Juang, B. H. (1986). An introduction to hidden Markov models. *ASSP Magazine*, 3(1):4–16.
